# Recognition of Manipulated Objects by Motor Learning

**Hiroaki Gomi**      **Mitsuo Kawato**
ATR Auditory and Visual Perception Research Laboratories,
Inui-dani, Sanpei-dani, Seika-cho, Soraku-gun, Kyoto 619-02, Japan

## Abstract

We present two neural network controller learning schemes based on *feedback-error-learning* and modular architecture for recognition and control of multiple manipulated objects. In the first scheme, a Gating Network is trained to acquire object-specific representations for recognition of a number of objects (or sets of objects). In the second scheme, an Estimation Network is trained to acquire function-specific, rather than object-specific, representations which directly estimate physical parameters. Both recognition networks are trained to identify manipulated objects using somatic and/or visual information. After learning, appropriate motor commands for manipulation of each object are issued by the control networks.

## 1 INTRODUCTION

Conventional feedforward neural-network controllers (Barto et al., 1983; Psaltis et al., 1987; Kawato et al., 1987, 1990; Jordan, 1988; Katayama & Kawato, 1991) can not cope with multiple or changeable manipulated objects or disturbances because they cannot change immediately the control law corresponding to the object. In interaction with manipulated objects or, in more general terms, in interaction with an environment which contains unpredictable factor, feedback information is essential for control and object recognition. From these considerations, Gomi & Kawato (1990) have examined the adaptive feedback controller learning schemes using *feedback-error-learning*, from which *impedance control* (Hogan, 1985) can be obtained automatically. However, in that scheme, some higher system needs to supervise the setting of the appropriate mechanical impedance for each manipulated object or environment.

In this paper, we introduce semi-feedforward control schemes using neural networks which receive feedback and/or feedforward information for recognition of multiple manipulated objects based on *feedback-error-learning* and modular network architecture. These schemes have two advantages over previous ones as follows. (1) Learning is achieved without the exact target motor command vector, which is unavailable during supervised motor learning. (2) Although somatic information alone was found to be sufficient to recognize objects, object identification is predictive and more reliable when both somatic and visual information are used.

## 2   RECOGNITION OF MANIPULATED OBJECTS

The most important issues in object manipulation are (1) how to recognize the manipulated object and (2) how to achieve uniform performance for different objects. There are several ways to acquire helpful information for recognizing manipulated objects. Visual information and somatic information (performance by motion) are most informative for object recognition for manipulation.

The physical characteristics useful for object manipulation such as mass, softness and slipperiness, can not be predicted without the experience of manipulating similar objects. In this respect, object recognition for manipulation should be learned through object manipulation.

## 3 MODULAR ARCHITECTURE USING GATING NETWORK

Jacobs et al. (1990, 1991) and Nowlan & Hinton (1990, 1991) have proposed a competitive modular network architecture which is applied to the task decomposition problem or classification problems. Jacobs (1991) applied this network architecture to the multi-payload robotics task in which each expert network controller is trained for each category of manipulated objects in terms of the object's mass. In his scheme, the payload's identity is fed to the gating network to select a suitable expert network which acts as a feedforward controller.

We examined modular network architecture using *feedback-error-learning* for simultaneous learning of object recognition and control task as shown in Fig.1.

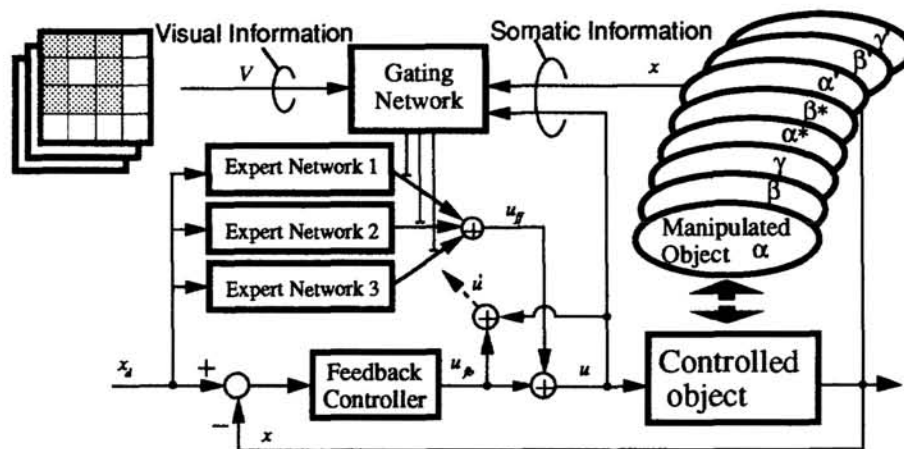

Fig.1   Configuration of the modular architecture using Gating Network
for object manipulation based on feedback-error-learning

In this learning scheme, the quasi-target vector for combined output of expert networks is employed instead of the exact target vector. This is because it is unlikely that the exact target motor command vector can be provided in learning. The quasi-target vector of feedforward motor command, $u'$ is produced by :

$$u' = u + u_{fb}. \tag{1}$$

Here, $u$ denotes the previous final motor command and $u_{fb}$ denotes the feedback motor command. Using this quasi-target vector, the gating and expert networks are trained to maximize the log-likelihood function, $\ln L$, by using backpropagation.

$$\ln L = \ln \sum_{i=1}^{n} g_i e^{-|u'-u_i|^2/2\sigma_i^2} \tag{2}$$

Here, $u_i$ is the $i$ th expert network output, $\sigma_i$ is a variance scaling parameter of the $i$ th expert network and $g_i$, the $i$ th output of gating network, is calculated by

$$g_i = \frac{e^{s_i}}{\sum_{j=1}^{n} e^{s_j}}, \tag{3}$$

where $s_i$ denotes the weighted input received by the $i$ th output unit. The total output of the modular network is

$$u_{ff} = \sum_{i=1}^{n} g_i u_i. \tag{4}$$

By maximizing Eq.2 using steepest ascent method, the gating network learns to choose the expert network whose output is closest to the quasi-target command, and each expert network is tuned correctly when it is chosen by the gating network. The desired trajectory is fed to the expert networks so as to make them work as feedforward controllers.

# 4 SIMULATION OF OBJECT MANIPULATION BY MODULAR ARCHITECTURE WITH GATING NETWORK

We show the advantage of the learning schemes presented above by simulation results below. The configuration of the controlled object and manipulated object is shown in Fig.2 in which $M$, $B$, $K$ respectively denote the mass, viscosity and stiffness of the coupled object (controlled- and manipulated-object). The manipulated object is changed every epoch (1 [sec]) while the coupled object is controlled to track the desired trajectory. Fig.3 shows the selected object, the feedforward and feedback motor commands, and the desired and actual trajectories before learning.

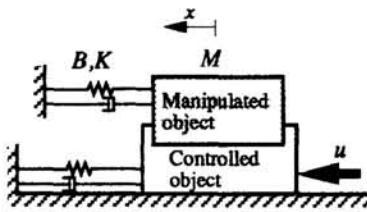

Fig.2 Configuration of the controlled object and the manipulated object

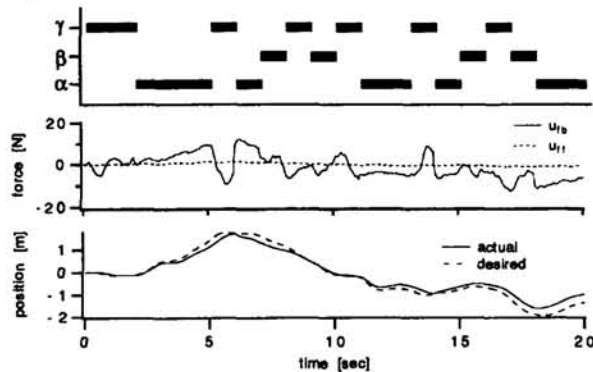

Fig.3 Temporal patterns of the selected object, the motor commands, the desired and actual trajectories before learning

The desired trajectory, $x_d$, was produced by Ornstein-Uhlenbeck random process. As shown in Fig.3, the error between the desired trajectory and the actual trajectory remained because the feedback controller in which the gains were fixed, was employed in this condition. (Physical characteristics of the objects used are listed in Fig.4a)

## 4.1 SOMATIC INFORMATION FOR GATING NETWORK

We call the actual trajectory vector, $x$, and the final motor command, $u$, "somatic information". Somatic information should be most useful for on-line (feedback) recognition of the dynamical characteristics of manipulated objects. The latest four times data of somatic information were used as the gating network inputs for identification of the coupled object in this simulation. $s$ of Eq.3 is expressed as:

$$s(t) = \psi_1(x(t), x(t-1), x(t-2), x(t-3), u(t), u(t-1), u(t-2), u(t-3)). \quad (5)$$

The dynamical characteristics of coupled objects are shown in Fig.4a. The object was changed in every epoch (1 [sec]). The variance scaling parameter was $\sigma_i = 0.8$ and the learning rates were $\eta_{gate} = 1.0 \times 10^{-3}$ and $\eta_{expert i} = 1.0 \times 10^{-5}$. The three-layered feedforward neural network (input 16, hidden 30, output 3) was employed for the gating network and the two-layered linear networks (input 3, output 1) were used for the expert networks.

Comparing the expert's weights after learning and the coupled object characteristics in Fig.4a, we realize that expert networks No.1, No.2, No.3 obtained the inverse dynamics of coupled objects $\gamma$, $\beta$, $\alpha$, respectively. The time variation of object, the gating network outputs, motor commands and trajectories after learning are shown in Fig.4b. The gating network outputs for the objects responded correctly in the most of the time and the feedback motor command, $u_{fb}$, was almost zero. As a consequence of adaptation, the actual trajectory almost perfectly corresponded with the desired trajectory.

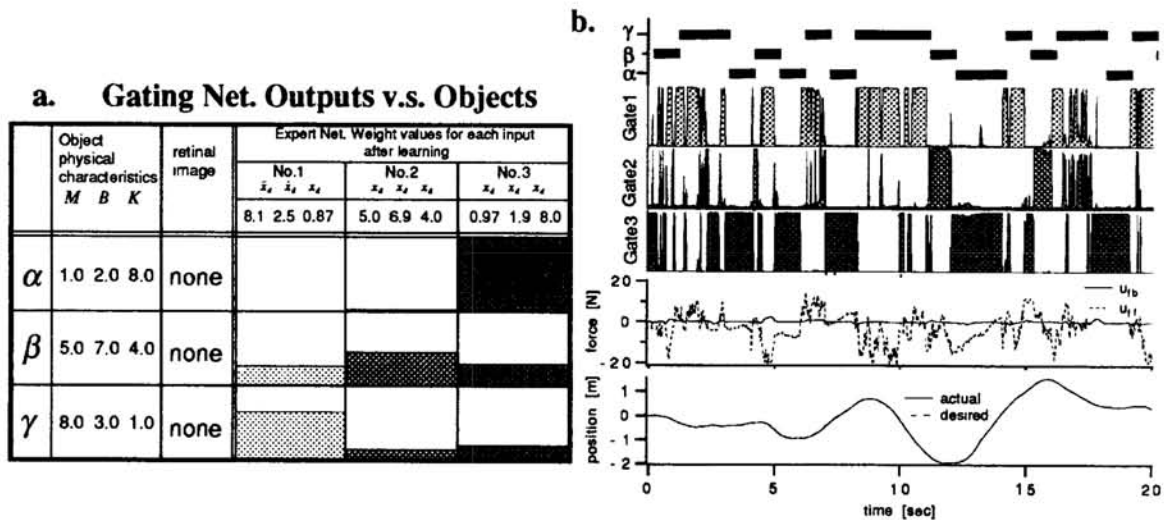

a. **Gating Net. Outputs v.s. Objects**

| Object physical characteristics $M$ $B$ $K$ | retinal image | Expert Net. Weight values for each input after learning | | |
|---|---|---|---|---|
| | | No.1 $\dot{x}_t$ $\ddot{x}_t$ $x_t$ | No.2 $x_t$ $\dot{x}_t$ $\ddot{x}_t$ | No.3 $x_t$ $\dot{x}_t$ $x_t$ |
| | | 8.1 2.5 0.87 | 5.0 6.9 4.0 | 0.97 1.9 8.0 |
| $\alpha$ 1.0 2.0 8.0 | none | | | |
| $\beta$ 5.0 7.0 4.0 | none | | | |
| $\gamma$ 8.0 3.0 1.0 | none | | | |

**Fig.4 Somatic information for gating network, a.** Statistical analysis of the correspondence of the expert networks with each object after learning (averaged gating outputs), **b.** Temporal patterns of objects, gating outputs, motor commands and trajectories after learning

## 4.2 VISUAL INFORMATION FOR GATING NETWORK

We usually assume the manipulated object's characteristics by using visual information. Visual information might be helpful for feedforward recognition. In this case, $s$ of Eq.3 is expressed as:

$$s(t) = \psi_2(V(t)) . \quad (6)$$

We used three visual cues corresponding to each coupled object in this simulation as shown in Fig.5a. At each epoch in this simulation, one of three visual cues selected randomly is randomly placed at one of four possible locations on a $4 \times 4$ retinal matrix.

The visual cues of each object are different, but object $\alpha$ and $\alpha^*$ have the same dynamical characteristics as shown in Fig.5a. The gating network should identify the object and select a suitable expert network for feedforward control by using this visual information. The learning coefficients were $\sigma_i = 0.7$, $\eta_{gate} = 1.0 \times 10^{-3}$, $\eta_{expert\ i} = 1.0 \times 10^{-5}$. The same networks used in above experiment were used in this simulation.

After learning, the expert network No.2 acquired the inverse dynamics of object $\alpha$ and $\alpha^*$, and expert network No.3 accomplished this for object $\gamma$. It is recognized from Fig.5b that the gating network almost perfectly selected expert network No.2 for object $\alpha$ and $\alpha^*$, and almost perfectly selected expert network No.3 for object $\gamma$. Expert network No.1 which did not acquire inverse dynamics corresponding to any of the three objects, was not selected in the test period after learning. The actual trajectory in the test period corresponded almost perfectly to the desired trajectory.

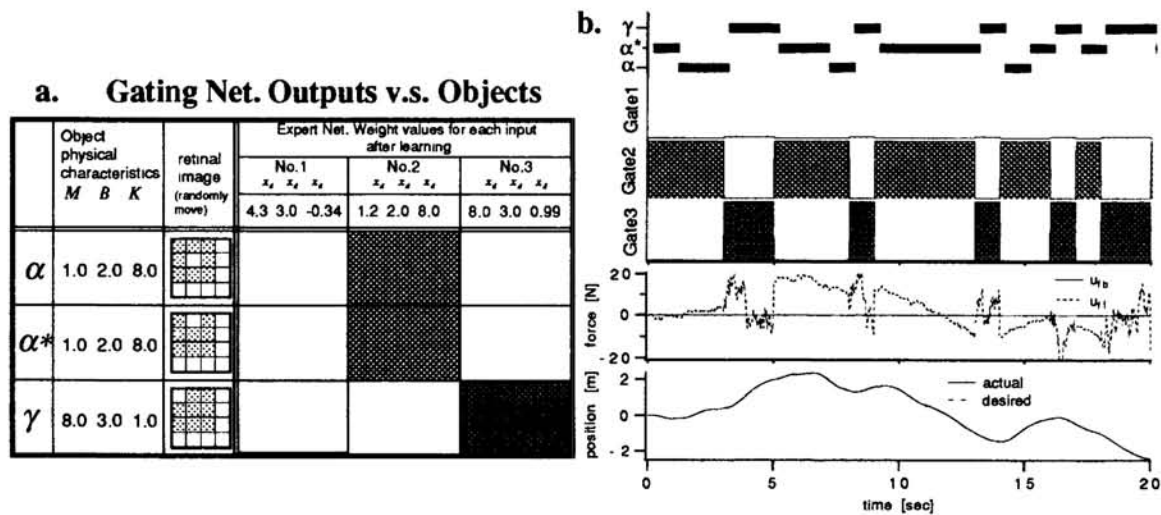

**Fig. 5  Visual information for gating network,  a.** Statistical analysis of the correspondence of the expert networks with each object after learning (averaged gating outputs), **b.** Temporal patterns of objects, gating outputs, motor commands and trajectories after learning

## 4.3 SOMATIC & VISUAL INFORMATION FOR GATING NETWORK

We show here the simulation results by using both of somatic and visual information as the gating network inputs. In this case, $s$ of Eq.3 is represented as:

$$s(t) = \psi_3\big(x(t), \cdots, x(t-3), u(t), \cdots, u(t-3), V(t)\big). \qquad (7)$$

In this simulation, the object $\alpha$ and $\beta^*$ had different dynamical characteristics, but shared same visual cue as listed in Fig.6a. Thus, to identify the coupled object one by one, it is necessary for the gating network to utilize not only visual information but also somatic information. The learning coefficients were $\sigma_i = 1.0$, $\eta_{gate} = 1.0 \times 10^{-3}$ and $\eta_{expert\ i} = 1.0 \times 10^{-5}$. The gating network had 32 input units, 50 hidden units, and 1 output unit, and the expert networks were the same as in the above experiment.

After learning, expert networks No.1, No.2, No.3 acquired the inverse dynamics of objects $\gamma$, $\beta^*$, $\alpha$ respectively. As shown in Fig.6b, the gating network identified the object almost correctly.

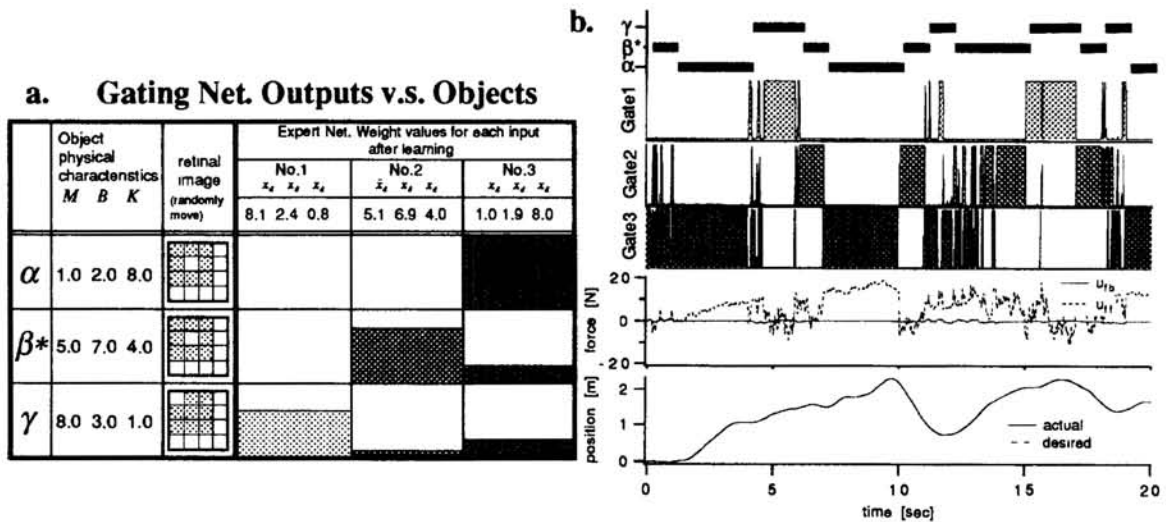

**a.    Gating Net. Outputs v.s. Objects**

| Object physical characteristics M B K | retinal image (randomly move) | Expert Net. Weight values for each input after learning | | |
|---|---|---|---|---|
| | | No.1 $x_d$ $x_d$ $x_d$ | No.2 $\dot{x}_d$ $x_d$ $x_d$ | No.3 $x_d$ $x_d$ $x_d$ |
| | | 8.1 2.4 0.8 | 5.1 6.9 4.0 | 1.0 1.9 8.0 |
| α | 1.0 2.0 8.0 | | | ■ |
| β* | 5.0 7.0 4.0 | | ▓ | |
| γ | 8.0 3.0 1.0 | ░ | | |

**Fig. 6 Somatic & Visual information for gating network, a.** Statistical analysis of the correspondence of the expert networks with each object after learning (averaged gating outputs), **b.** Temporal patterns of objects, gating outputs, motor commands and trajectories after learning

## 4.4 UNKNOWN OBJECT RECOGNITION BY USING SOMATIC INFORMATION

Fig.7b shows the responses for unknown objects whose physical characteristics were slightly different from known objects (see Fig.7a and Fig.4a) in the case using somatic information as the gating network inputs. Even if each tested object was not the same as any of the known (learned) objects, the closest expert network was selected. (compare Fig.4a and Fig.7a) During some period in the test phase, the feedback command increased because of an inappropriate feedforward command.

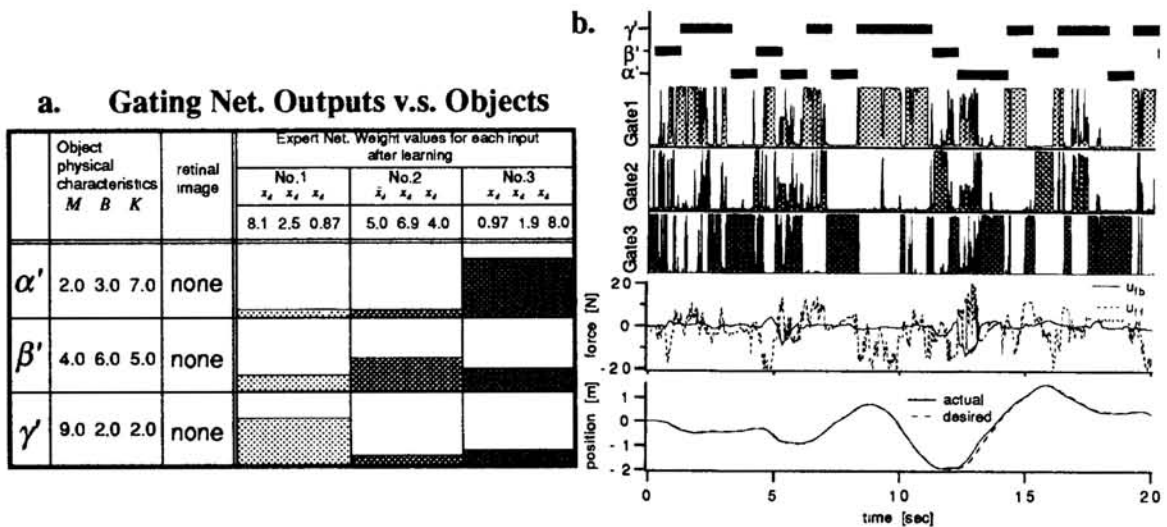

**a.    Gating Net. Outputs v.s. Objects**

| Object physical characteristics M B K | retinal image | Expert Net. Weight values for each input after learning | | |
|---|---|---|---|---|
| | | No.1 $x_d$ $x_d$ $x_d$ | No.2 $\dot{x}_d$ $x_d$ $x_d$ | No.3 $x_d$ $x_d$ $x_d$ |
| | | 8.1 2.5 0.87 | 5.0 6.9 4.0 | 0.97 1.9 8.0 |
| α' | 2.0 3.0 7.0 | none | | ■ |
| β' | 4.0 6.0 5.0 | none | ▓ | |
| γ' | 9.0 2.0 2.0 | none | | |

**Fig. 7 Unknown objects recognition by using Somatic information, a.** Statistical analysis of the correspondence of the expert networks with each object after learning (averaged gating outputs), **b.** Temporal patterns of objects, gating outputs, motor commands and trajectories after learning

# 5 MODULAR ARCHITECTURE
# USING ESTIMATION NETWORK

The previous modular architecture is competitive in the sense that expert networks compete with each other to occupy its niche in the input space. We here propose a new cooperative modular architecture where expert networks specified for different functions cooperate to produce the required output. In this scheme, estimation networks are trained to recognize physical characteristics of manipulated objects by using feedback information. Using this method, an infinite number of manipulated objects in the limited domain can be treated by using a small number of estimation networks. We applied this method to recognizing the mass of the manipulated objects. (see Fig.8)

Fig.9a shows the output of the estimation network compared to actual masses. The realized trajectory almost coincided with the desired trajectory as shown in Fig.9b. This learning scheme can be applied not only to estimating mass but also to other physical characteristics such as softness or slipperiness.

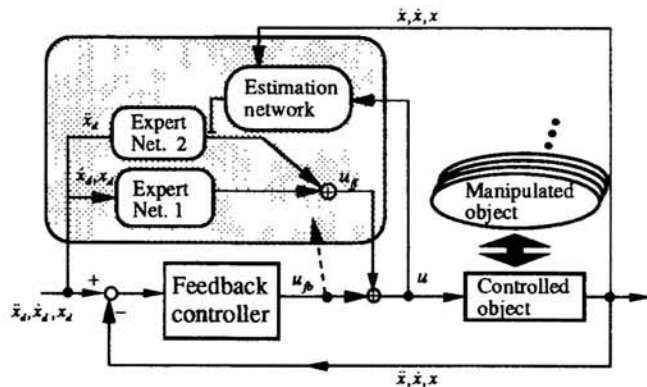

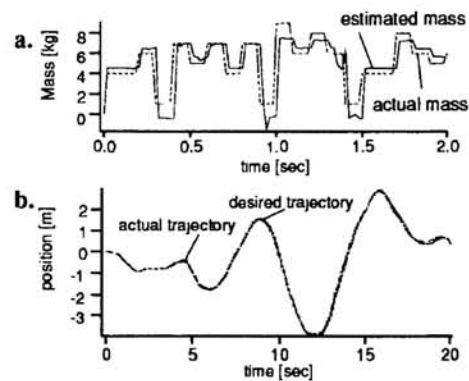

Fig. 8   Configuration of the modular architecture using mass estimation network for object manipulation by feedback-error-learning

Fig. 9 a. Comparison of actual & estimated mass, b. desired & actual trajectory

# 6 DISCUSSION

In the first scheme, the internal models for object manipulation (in this case, inverse dynamics) were represented not in terms of visual information but rather, of somatic information (see 4.2). Although the current simulation is primitive, it indicates the very important issue that functional internal-representations of objects (or environments), rather than declarative ones, were acquired by motor learning.

The quasi-target motor command in the first scheme and the motor command error in the second scheme are not always exactly correct in each time step because the proposed learning schemes are based on the *feedback-error-learning* method. Thus, the learning rates in the proposed schemes should be slower than those schemes in which exact target commands are employed. In our preliminary simulation, it was about five times slower. However, we emphasize that exact target motor commands are not available in supervised motor learning.

The limited number of controlled objects which can be dealt with by the modular network with a gating network is a considerable problem (Jacobs, 1991; Nowlan, 1990, 1991). This problem depends on choosing an appropriate number of expert networks and value of the variance scaling parameter, $\sigma$. Once this is done, the expert networks can interpolate

the appropriate output for a number of unknown objects. Our second scheme provides a more satisfactory solution to this problem.

On the other hand, one possible drawback of the second scheme is that it may be difficult to estimate many physical parameters for complicated objects, even though the learning scheme which directly estimates the physical parameters can handle any number of objects.

We showed here basic examinations of two types of neural networks - a gating network and a direct estimation network. Both networks use feedback and/or feedforward information for recognition of multiple manipulated objects. In future, we will attempt to integrate these two architectures in order to model tasks involving skilled motor coordination and high level recognition.

## Acknowledgment

We would like to thank Drs. E. Yodogawa and K. Nakane of ATR Auditory and Visual Perception Research Laboratories for their continuing encouragement. Supported by HFSP Grant to M.K.

# References

Barto, A.G., Sutton, R.S., Anderson, C.W. (1983) Neuronlike adaptive elements that can solve difficult learning control problems; *IEEE Trans. on Sys. Man and Cybern.* SMC-13, pp.834-846

Gomi, H., Kawato, M. (1990) Learning control for a closed loop system using feedback-error-learning. *Proc. the 29th IEEE Conference on Decision and Control*, Hawaii, Dec., pp.3289-3294

Hogan, N. (1985) Impedance control: An approach to manipulation: Part I - Theory, Part II - Implementation, Part III - Applications, *ASME Journal of Dynamic Systems, Measurement*, and Control, Vol.107, pp.1-24

Jacobs, R.A., Jordan, M.I., Barto, A.G. (1990) Task decomposition through competition in a modular connectionist architecture: The what and where vision tasks, *COINS Technical Report 90-27*, pp.1-49

Jacobs, R.A., Jordan, M.I. (1991) A competitive modular connectionist architecture. In Lippmann, R.P. et al., (Eds.) *NIPS 3*, pp.767-773

Jordan, M.I. (1988) Supervised learning and systems with excess degrees of freedom, *COINS Technical Report 88-27*, pp.1-41

Kawato, M., Furukawa, K., Suzuki, R. (1987) A hierarchical neural-network model for control and learning of voluntary movement; *Biol. Cybern. 57*, pp.169-185

Kawato, M. (1990) Computational schemes and neural network models for formation and control of multijoint arm trajectory. In: Miller, T., Sutton, R.S., Werbos, P.J.(Eds.) *Neural Networks for Control*, The MIT Press, Cambridge, Massachusetts, pp.197-228

Katayama, M., Kawato, M. (1991) Learning trajectory and force control of an artificial muscle arm by parallel-hierarchical neural network model. In Lippmann, R.P. et al., (Eds.) *NIPS 3*, pp.436-442

Nowlan, S.J. (1990) Competing experts: An experimental investigation of associative mixture models, *Univ. Toronto Tech. Rep. CRG-TR-90-5*, pp.1-77

Nowlan, S.J., Hinton, G.E. (1991) Evaluation of adaptive mixtures of competing experts. In Lippmann, R.P. et al., (Eds.) *NIPS 3*, pp.774-780

Psaltis, D., Sideris, A., Yamamura, A. (1987) Neural controllers, *Proc. IEEE Int. Conf. Neural Networks*, Vol.4, pp.551-557